# Mining Internet-Scale Software Repositories

**Erik Linstead, Paul Rigor, Sushil Bajracharya, Cristina Lopes and Pierre Baldi**
Donald Bren School of Information and Computer Science
University of California, Irvine
Irvine, CA 92697-3435
{elinstea,prigor,sbajrach,lopes,pfbaldi}@ics.uci.edu

## Abstract

Large repositories of source code create new challenges and opportunities for statistical machine learning. Here we first develop Sourcerer, an infrastructure for the automated crawling, parsing, and database storage of open source software. Sourcerer allows us to gather Internet-scale source code. For instance, in one experiment, we gather 4,632 java projects from SourceForge and Apache totaling over 38 million lines of code from 9,250 developers. Simple statistical analyses of the data first reveal robust power-law behavior for package, SLOC, and lexical containment distributions. We then develop and apply unsupervised author-topic, probabilistic models to automatically discover the topics embedded in the code and extract topic-word and author-topic distributions. In addition to serving as a convenient summary for program function and developer activities, these and other related distributions provide a statistical and information-theoretic basis for quantifying and analyzing developer similarity and competence, topic scattering, and document tangling, with direct applications to software engineering. Finally, by combining software textual content with structural information captured by our CodeRank approach, we are able to significantly improve software retrieval performance, increasing the AUC metric to 0.84– roughly 10-30% better than previous approaches based on text alone. Supplementary material may be found at: http://sourcerer.ics.uci.edu/nips2007/nips07.html.

## 1 Introduction

Large repositories of private or public software source code, such as the open source projects available on the Internet, create considerable new opportunities and challenges for statistical machine learning, information retrieval, and software engineering. Mining such repositories is important, for instance, to understand software structure, function, complexity, and evolution, as well as to improve software information retrieval systems and identify relationships between humans and the software they produce. Tools to mine source code for functionality, structural organization, team structure, and developer contributions are also of interest to private industry, where these tools can be applied to such problems as in-house code reuse and project staffing. While some progress has been made in the application of statistics and machine learning techniques to mine software corpora, empirical studies have typically been limited to small collections of projects, often on the order of one hundred projects or less, several orders of magnitude smaller than publicly available repositories(eg. [1]).

Mining large software repositories requires leveraging both the textual and structural aspects of software data, as well as any relevant meta data. Here we develop Sourcerer, a large-scale infrastructure to explore such aspects. We first identify a number of robust power-law behaviors by simple statistical analyses. We then develop and apply unsupervised author-topic probabilistic models to discover the topics embedded in the code and extract topic-word and author-topic distributions. Finally, we leverage the dual textual and graphical nature of software to improve code search and retrieval.

## 2 Infrastructure and Data

To allow for the Internet-scale analysis of source code we have built Sourcerer, an extensive infrastructure designed for the automated crawling, downloading, parsing, organization, and storage of large software repositories in a relational database. A highly configurable crawler allows us to specify the number and types of projects desired, as well as the host databases that should be targeted, and to proceed with incremental updates in an automated fashion. Once target projects are downloaded, a depackaging module uncompresses archive files while saving useful metadata (project name, version, etc). While the infrastructure is general, we apply it here to a sample of projects in Java. Specifically, for the results reported, we download 12,151 projects from Sourceforge and Apache and filter out distributions packaged without source code (binaries only). The end result is a repository consisting of 4,632 projects, containing 244,342 source files, with 38.7 million lines of code, written by 9,250 developers. For the software author-topic modeling approach we also employ the Eclipse 3.0 source code as a baseline. Though only a single project, Eclipse is a large, active open source effort that has been widely studied. In this case, we consider 2,119 source files, associated with about 700,000 lines of code, a vocabulary of 15,391 words, and 59 programmers. Methods for extracting and assigning words and programmers to documents are described in the next sections. A complete list of all the projects contained in our repository is available from the supplementary materials web pages.

## 3 Statistical Analysis

During the parsing process our system performs a static analysis on project source code files to extract code entities and their relationships, storing them in a relational database. For java these entities consist of packages, classes, interfaces, methods, and fields, as well as more specific constructs such as constructors and static initializers. Relations capture method calls, inheritance, and encapsulation, to name a few. The populated database represents a substantial foundation on which to base statistical analysis of source code. Parsing the multi-project repository described above yields a repository of over 5 million entities organized into 48 thousand packages, 560 thousand classes, and 3.2 million methods, participating in over 23.4 million relations. By leveraging the query capabilities of the underlying database we can investigate other interesting statistics. For example, table 1 contains the frequencies of Java keywords across all 4,632 projects. Upon examining this data we can see that the 'default' keyword occurs about 6 percent less frequently than the 'switch' keyword, despite the fact that best practice typically mandates all switch statements contain a default block. Moreover, the 'for' loop is about twice as pervasive as the 'while' loop, suggesting that the bound on the number of iterations is more likely to be known or based on the size of a known data structure.

Table 1: Frequency of java keyword occurrence

| Keyword | Percentage | Keyword | Percentage | Keyword | Percentage | Keyword | Percentage |
|---------|------------|---------|------------|---------|------------|---------|------------|
| public | 12.53 | boolean | 2.12 | this | 0.89 | switch | 0.19 |
| if | 8.44 | false | 1.69 | break | 0.85 | interface | 0.17 |
| new | 8.39 | case | 1.60 | while | 0.63 | continue | 0.15 |
| return | 7.69 | true | 1.60 | super | 0.57 | finally | 0.14 |
| import | 6.89 | class | 1.36 | instanceof | 0.56 | default | 0.13 |
| int | 6.54 | protected | 1.33 | double | 0.55 | native | 0.08 |
| null | 5.52 | catch | 1.33 | long | 0.54 | transient | 0.06 |
| void | 4.94 | for | 1.22 | implements | 0.43 | do | 0.05 |
| private | 3.66 | try | 1.22 | char | 0.30 | assert | 0.03 |
| static | 3.16 | throw | 1.16 | float | 0.28 | enum | 0.02 |
| final | 3.01 | package | 0.96 | abstract | 0.25 | volatile | 0.04 |
| else | 2.33 | byte | 0.93 | synchronized | 0.25 | strictfp | 2.49E-06 |
| throws | 2.16 | extends | 0.89 | short | 0.20 | | |

Finally, statistical analyses of distributions also identify several power-law distributions. We have observed power-law distributions governing package, SLOC, and inside relation (lexical contain-

ment) counts. For instance, Figure 1 shows the log-log plots for the number of packages across projects. Similar graphs for other distributions are available from the supplemental materials page.

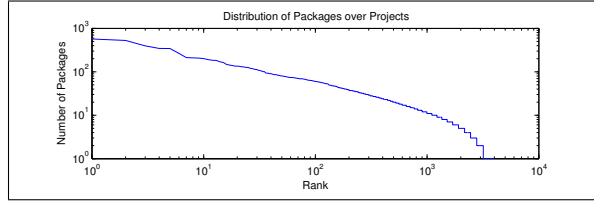

Figure 1: Approximate power-law distribution for packages over projects

# 4 Topic and Author-Topic Probabilistic Modeling of Source Code

Automated topic and author-topic modeling have been successfully used in text mining and information retrieval where they have been applied, for instance, to the problem of summarizing large text corpora. Recent techniques include Latent Dirichlet Allocation (LDA), which probabilistically models text documents as mixtures of latent topics, where topics correspond to key concepts presented in the corpus [2] (see also [3]). Author-Topic (AT) modeling is an extension of topic modeling that captures the relationship of authors to topics in addition to extracting the topics themselves. An extension of LDA to probabilistic AT modeling has been developed in [4]. In the literature [5], these more recent approaches have been found to produce better results than more traditional methods such as latent semantic analysis (LSA) [6]. Despite previous work in classifying code based on concepts [1], applications of LDA and AT models have been limited to traditional text corpora such as academic publications, news reports, corporate emails, and historical documents [7, 8]. At the most basic level, however, a code repository can be viewed as a text corpus, where source files are analogous to documents and developers to authors. Though vocabulary, syntax, and conventions differentiate a programming language from a natural language, the tokens present in a source file are still indicative of its function (ie. its topics). Thus here we develop and apply probabilistic AT models to software data.

In AT models for text, the data consists of a set of documents. The authors of each documents are known and each document is treated as a bag of words. We let $A$ be the total number of authors, $W$ the total number of distinct words (vocabulary size), and $T$ the total number of topics present in the documents. While non-parametric Bayesian [9] and other [10] methods exist to try to infer $T$ from the data, here we assume that $T$ is fixed (e.g. $T = 100$), though we explore different values.

As in [7], our model assumes that each topic $t$ is associated with a multinomial distribution $\phi_{\bullet t}$ over words $w$, and each author $a$ is associated with a multinomial distribution $\theta_{\bullet a}$ over topics. More precisely, the parameters are given by two matrices: a $T \times A$ matrix $\Theta = (\theta_{ta})$ of author-topic distributions, and a $W \times T$ matrix $\Phi = (\phi_{wt})$ of topic-word distributions. Given a document $d$ containing $N_d$ words with known authors, in generative mode each word is assigned to one of the authors $a$ of the document uniformly, then the corresponding $\theta_{\bullet a}$ is sampled to derive a topic $t$, and finally the corresponding $\phi_{\bullet t}$ is sampled to derive a word $w$. A fully Bayesian model is derived by putting symmetric Dirichlet priors with hyperparameters $\alpha$ and $\beta$ over the distributions $\theta_{\bullet a}$ and $\phi_{\bullet t}$. So for instance the prior on $\theta_{\bullet a}$ is given by

$$D_\alpha(\theta_{\bullet a}) = \frac{\Gamma(T\alpha)}{(\Gamma(\alpha))^T} \prod_{t=1}^{T} \theta_{ta}^{\alpha-1}$$

and similarly for $\phi_{\bullet t}$. If $\mathcal{A}$ is the set of authors of the corpus and document $d$ has $A_d$ authors, it is easy to see that under these assumptions the likelihood of a document is given by:

$$P(d|\Theta, \Phi, \mathcal{A}) = \prod_{i=1}^{N_d} \frac{1}{A_d} \sum_a \sum_{t=1}^{T} \phi_{w_i t} \theta_{ta}$$

which can be integrated over $\phi$ and $\theta$ and their Dirichlet distributions to get $P(d|\alpha, \beta, \mathcal{A})$. The posterior can be sampled efficiently using Markov Chain Monte Carlo Methods (Gibbs sampling) and, for instance, the $\Theta$ and $\Phi$ parameter matrices can be estimated by MAP or MPE methods.

Once the data is obtained, applying this basic AT model to software requires the development of several tools to facilitate the processing and modeling of source code. In addition to the crawling infrastructure described above, the primary functions of the remaining tools are to extract and resolve author names from source code, as well as convert the source code to the bag-of-words format.

## 4.1 Information Extraction from Source Code

**Author-Document:** The author-document matrix is produced from the output of our author extraction tool. It is a binary matrix where entry $[i,j]$=1 if author $i$ contributed to document $j$, and 0 otherwise. Extracting author information is ultimately a matter of tokenizing the code and associating developer names with file (document) names when this information is available. This process is further simplified for java software due to the prevalence of javadoc tags which present this metadata in the form of attribute-value pairs.

Exploratory analysis of the Eclipse 3.0 code base, however, shows that most source files are credited to "The IBM Corporation" rather than specific developers. Thus, to generate a list of authors for specific source files, we parsed the Eclipse bug data available in [11]. After pruning files not associated with any author, this input dataset consists of 2,119 Java source files, comprising 700,000 lines of code, from a total of 59 developers.

While leveraging bug data is convenient (and necessary) to generate the developer list for Eclipse 3.0, it is also desirable to develop a more flexible approach that uses only the source code itself, and not other data sources. Thus to extract author names from source code we also develop a lightweight parser that examines the code for javadoc '@author' tags, as well as free form labels such as 'author' and 'developer.' Occurrences of these labels are used to isolate and identify developer names. Ultimately author identifiers may come in the form of full names, email addresses, url's, or CVS account names. This multitude of formats, combined with the fact that author names are typically labeled in the code header, is key to our decision to extract developer names using our own parsing utilities, rather than part-of-speech taggers [12] leveraged in other text mining projects.

A further complication for author name extraction is the fact that the same developer may write his name in several different ways. For example, "John Q. Developer" alternates between "John Developer," "J. Q. Developer," or simply "Developer." To account for this effect, we implement also a two-tiered approach to name resolution using the q-gram algorithm [13]. When an individual project is parsed, a list of contributing developers (and the files they modified) is created. A pairwise comparison of author-names is then performed using q-gram similarity, and pairs of names whose similarity is greater than a threshold $t1$ are merged. This process continues until all pairwise similarities are below the threshold, and the project list is then added to a global list of authors. When parsing is complete for all projects, the global author list is resolved using the same process, but with a new threshold, $t2$, such that $t2 > t1$. This approach effectively implements more conservative name resolution across projects in light of the observation that the scope of most developer activities is limited to a relatively small number (1 in many cases) of open source efforts. In practice, we set $t1 = .65$ and $t2 = .75$. Running our parser on the multi-project repository yields 9,250 distinct authors respectively.

**Word-Document:** To produce the word-document matrix for our input data we have developed a comprehensive tokenization tool tuned to the Java programming language. This tokenizer includes language-specific heuristics that follow the commonly practiced naming conventions. For example, the Java class name "QuickSort" will generate the words "quick" and "sort". All punctuation is ignored. As an important step in processing source files our tool removes commonly occurring stop words. We augment a standard list of stop words used for the English language (e.g. and, the, but, etc) to include the names of all classes from the Java SDK (eg. ArrayList, HashMap, etc). This is done to specifically avoid extracting common topics relating to the Java collections framework.We run the LDA-based AT algorithm on the input matrices and set the total number of topics (100) and the number of iterations by experimentation. For instance, the number of iterations, $i$, to run the algorithm is determined empirically by analyzing results for $i$ ranging from 500 to several thousands. The results presented in the next section are derived using 3,000 iterations, which were found to

produce interpretable topics in a reasonable amount of time (a week or so). Because the algorithm contains a stochastic component we also verified the stability of the results across multiple runs.

## 4.2 Topic and Author-Topic Modeling Results

A representative subset of 6 topics extracted via Author-Topic modeling on the selected 2,119 source files from Eclipse 3.0 is given in Table 2. Each topic is described by several words associated with the topic concept. To the right of each topic is a list of the most likely authors for each topic with their probabilities. Examining the topic column of the table it is clear that various functions of the Eclipse framework are represented. For example, topic 1 clearly corresponds to unit testing, topic 2 to debugging, topic 4 to building projects, and topic 6 to automated code completion. Remaining topics range from package browsing to compiler options.

Table 2: Representative topics and authors from Eclipse 3.0

| # | Topic | Author Probabilities | # | Topic | Author Probabilities |
|---|---|---|---|---|---|
| 1 | junit<br>run<br>listener<br>item<br>suite | egamma 0.97065<br>wmelhem 0.01057<br>darin 0.00373<br>krbarnes 0.00144<br>kkolosow 0.00129 | 4 | nls-1<br>ant<br>manager<br>listener<br>classpath | darins 0.99572<br>dmegert 0.00044<br>nick 0.00044<br>kkolosow 0.00036<br>maeschli 0.00031 |
| 2 | target<br>source<br>debug<br>breakpoint<br>location | jaburns 0.96894<br>darin 0.02101<br>lbourlier 0.00168<br>darins 0.00113<br>jburns 0.00106 | 5 | type<br>length<br>names<br>match<br>methods | kjohnson 0.59508<br>jlanneluc 0.32046<br>darin 0.02286<br>johna 0.00932<br>pmulet 0.00918 |
| 3 | ast<br>button<br>cplist<br>entries<br>astnode | maeschli 0.99161<br>mkeller 0.00097<br>othomann 0.00055<br>tmaeder 0.00055<br>teicher 0.00046 | 6 | token<br>completion<br>current<br>identifier<br>assist | daudel 0.99014<br>teicher 0.00308<br>jlanneluc 0.00155<br>twatson 0.00084<br>dmegert 0.00046 |

Table 3 presents 6 representative author-topic assignments from the multi-project repository. This dataset yields a substantial increase in topic diversity. Topics representing major sub-domains of software development are clearly represented, with the first topic corresponding to web applications, the second to databases, the third to network applications, and the fourth to file processing. Topics 5 and 6 are especially interesting, as they correspond to common examples of crosscutting concerns from aspect-oriented programming [14], namely security and logging. Topic 5 is also demonstrative of the inherent difficulty of resolving author names, and the shortcomings of the q-gram algorithm, as the developer "gert van ham" and the developer "hamgert" are most likely the same person documenting their name in different ways.

Several trends reveal themselves when all results are considered. Though the majority of topics can be intuitively mapped to their corresponding domains, some topics are too noisy to be able to associate any functional description to them. For example, one topic extracted from our repository consists of Spanish words unrelated to software engineering which seem to represent the subset of source files with comments in Spanish. Other topics appear to be very project specific, and while they may indeed describe a function of code, they are not easily understood by those who are only casually familiar with the software artifacts in the codebase. This is especially true with Eclipse, which is limited in both the number and diversity of source files. In general noise appears to diminish as repository size grows. Noise can be controlled to some degree with tuning the number of topics to be extracted, but of course can not be eliminated completely.

Examining the author assignments (and probabilities) for the various topics provides a simple means by which to discover developer contributions and infer their competencies. It should come as no surprise that the most probable developer assigned to the JUnit framework topic is "egamma", or Erich Gamma. In this case, there is a 97% chance that any source file in our dataset assigned to this topic will have him as a contributor. Based on this rather high probability, we can also infer that he is likely to have extensive knowledge of this topic. This is of course a particularly

Table 3: Representative topics and authors from the multi-project repository

| # | Topic | Author Probabilities | # | Topic | Author Probabilities |
|---|---|---|---|---|---|
| 1 | servlet<br>session<br>response<br>request<br>http | craig r mcclanahan 0.19147<br>remy maucherat 0.08301<br>peter rossbach 0.04760<br>greg wilkins 0.04251<br>amy roh 0.03100 | 4 | file<br>path<br>dir<br>directory<br>stream | adam murdoch 0.02466<br>peter donald 0.02056<br>ludovic claude 0.01496<br>matthew hawthorne 0.01170<br>lk 0.01106 |
| 2 | sql<br>column<br>jdbc<br>type<br>result | mark matthews 0.33265<br>ames 0.02640<br>mike bowler 0.02033<br>manuel laflamme 0.02027<br>gavin king 0.01813 | 5 | token<br>key<br>security<br>param<br>cert | werner dittmann 0.09409<br>apache software foundation 0.06117<br>gert van ham 0.05153<br>hamgert 0.05144<br>jcetaglib.sourceforge.net 0.05133 |
| 3 | packet<br>type<br>session<br>snmpwalkmv<br>address | brian weaver 0.14015<br>apache directory project 0.10066<br>opennms 0.08667<br>matt whitlock 0.06508<br>trustin lee 0.04752 | 6 | service<br>str<br>log<br>config<br>result | wayne m osse 0.44638<br>dirk mascher 0.07339<br>david irwin 0.04928<br>linke 0.02823<br>jason 0.01505 |

attractive example because Erich Gamma is widely known for being a founder of the JUnit project, a fact which lends credibility to the ability of the topic modeling algorithm to assign developers to reasonable topics. One can interpret the remaining author-topic assignments along similar lines. For example, developer "daudel" is assigned to the topic corresponding to automatic code completion with probability .99. Referring back to the Eclipse bug data it is clear that the overwhelming majority of bug fixes for the codeassist framework were made by this developer. One can infer that this is likely to be an area of expertise of the developer.

In addition to determining developer contributions, one may also be curious to know the scope of a developer's involvement. Does a developer work across application areas, or are his contributions highly focused? How does the breadth of one developer compare to another? These are natural questions that arise in the software development process. To answer these questions within the framework of author-topic models, we can measure the breadth of an author $a$ by the entropy $H(a) = -\sum_t \theta_{ta} \log \theta_{ta}$ of the corresponding distribution over topics. Applying the measure to our multi-project dataset, we find that the average measure is 2.47 bits. The developer with the lowest entropy is "thierry danard," with .00076 bits. The developer with the highest entropy is "wdi" with 4.68 bits, with 6.64 bits being the maximum possible score for 100 topics. While the entropy

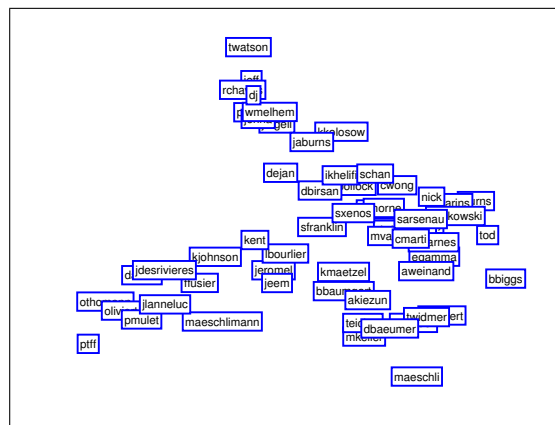

Figure 2: All 59 Eclipse 3.0 authors clustered by KL divergence

of the distribution of an author over topics measures the author's breadth, the similarity between two authors can be measured by comparing their respective distributions over topics. Several metrics are possible for this purpose, but one of the most natural measures is provided by the symmetrized Kullback-Leibler (KL) divergence. Multidimensional scaling (MDS) is employed to further visual-

ize author similarities, resulting in Figure 2 for the Eclipse project. The boxes represent individual developers, and are arranged such that developers with similar topic distributions are nearest one another. A similar figure, displaying only a subset of the 4,500 SourceForge and Apache authors due to space and legibility constraints, is available in the supplementary materials. This information is especially useful when considering how to form a development team, choosing suitable programmers to perform code updates, or deciding to whom to direct technical questions. Two other important distributions that can be retrieved from the AT modeling approach are the distribution of topics across documents, and the distribution of documents across topics (not shown). The corresponding entropies provide an automated and novel way to precisely formalize and measure topic scattering and document tangling, two fundamental concepts of software design [14], which are important to software architects when performing activities such as code refactoring.

## 5  Code Search and Retrieval

Sourcerer relies on a deep analysis of code to extract pertinent textual and structural features that can be used to improve the quality and performance of source code search, as well as augment the ways in which code can be searched. By combining standard text information retrieval techniques with source-specific heuristics and a relational representation of code, we have available a comprehensive platform for searching software components. While there has been progress in developing source-code-specific search engines in recent years (e.g. Koders, Krugle, and Google's CodeSearch), these systems continue to focus strictly on text information retrieval, and do not appear to leverage the copious relations that can be extracted and analyzed from code.

Programs are best modeled as graphs, with code entities comprising the nodes and various relations the edges. As such, it is worth exploring possible ranking methods that leverage the underlying graphs. A natural starting point is Google's PageRank [15], which considers hyperlinks to formulate a notion of popularity among web pages. This can be applied to source as well, as it is likely that a code entity referenced by many other entities are more robust than those with few references.

We used Google's PageRank [15] almost verbatim. The Code Rank of a code entity (package, class, or method) $A$ is given by: $CR(A) = (1 - d) + d(CR(T_1)/C(T_1) + ... + CR(T_n)/C(T_n))$ where $T_1...T_n$ are the code entities referring to $A$, $C(A)$ is the number of outgoing links of $A$, and $d$ is a damping factor.

Using the CodeRank algorithm as a basis it is possible to devise many ranking schemes by building graphs from the many entities and relations stored in our database, or subsets thereof. For example, one may consider the graph of only method call relationships, package dependencies, or inheritance hierarchies. Moreover, graph-based techniques can be combined with a variety of heuristics to further improve code search. For example, keyword hits to the right of the fully-qualified name can be boosted, hits in comments can be discounted, and terms indicative of test articles can be ignored.

We are conducting detailed experiments to assess the effectiveness of graph-based algorithms in conjunction with standard IR techniques to search source code. Current evidence strongly indicates that best results are ultimately obtained by combining term-based ranking with source-specific heuristics and coderank. After defining a set of 25 control queries with known "best" hits, we compared performances using standard information retrieval metrics, such as area under curve (AUC). Queries were formulated to represent users searching for specific algorithms, such as 'depth first search,' as well as users looking to reuse complete components, such as 'database connection manager.' Best hits were determined manually with a team of 3 software engineers serving as human judges of result quality, modularity, and ease of reuse. Results clearly indicate that the general Google search engine is ineffective for locating relevant source code, with a mean AUC of .31 across the queries. By restricting its corpus to code alone, Google's code search engine yields substantial improvement with an AUC of approximately .66. Despite this improvement this system essentially relies only on regular expression matching of code keywords. Using a Java-specific keyword and comment parser our infrastructure yields an immediate improvement with an AUC of .736. By augmenting this further with the heuristics above and CodeRank (consisting of class and method relations), the mean AUC climbs to .841. At this time we have conducted extensive experiments for 12 ranking schemes corresponding to various combinations of graph-based and term-based heuristics, and have observed similar improvements. While space does not allow their inclusion, additional results are available from our supplementary materials page.

# 6 Conclusion

Here we have leveraged a comprehensive code processing infrastructure to facilitate the mining of large-scale software repositories. We conduct a statistical analysis of source code on a previously unreported scale, identifying robust power-law behavior among several code entities. The development and application of author-topic probabilistic modeling to source code allows for the unsupervised extraction of program organization, functionality, developer contributions, and developer similarities, thus providing a new direction for research in this area of software engineering. The methods developed are applicable at multiple scales, from single projects to Internet-scale repositories. Results indicate that the algorithm produces reasonable and interpretable automated topics and author-topic assignments. The probabilistic relationships between author, topics, and documents that emerge from the models naturally provide an information-theoretic basis to define and compare developer and program similarity, topic scattering, and document tangling with potential applications in software engineering ranging from bug fix assignments and staffing to software refactoring. Finally, by combining term-based information retrieval techniques with graphical information derived from program structure, we are able to significantly improve software search and retrieval performance.

**Acknowledgments:**  Work in part supported by NSF MRI grant EIA-0321390 and a Microsoft Faculty Research Award to PB, as well as NSF grant CCF-0725370 to CL and PB.

# References

[1] S. Ugurel, R. Krovetz, and C. L. Giles. What's the code?: automatic classification of source code archives. In *KDD '02: Proceedings of the eighth ACM SIGKDD international conference on Knowledge discovery and data mining*, pages 632–638, New York, NY, USA, 2002. ACM Press.

[2] D.M. Blei, A.Y. Ng, and M.I. Jordan. Latent dirichlet allocation. *Journal of Machine Learning Research*, 3:993–1022, January 2003.

[3] W. Buntine. Open source search: a data mining platform. *SIGIR Forum*, 39(1):4–10, 2005.

[4] M. Steyvers, P. Smyth, M. Rosen-Zvi, and T. Griffiths. Probabilistic author-topic models for information discovery. In *KDD '04: Proceedings of the tenth ACM SIGKDD international conference on Knowledge discovery and data mining*, pages 306–315, New York, NY, USA, 2004. ACM Press.

[5] D. Newman, C. Chemudugunta, P. Smyth, and M. Steyvers. Analyzing entities and topics in news articles using statistical topic models. In *ISI*, pages 93–104, 2006.

[6] S. Deerwester, S. Dumais, T. Landauer, G. Furnas, and R. Harshman. Indexing by latent semantic analysis. *Journal of the American Society of Information Science*, 41(6):391–407, 1990.

[7] Michal Rosen-Zvi, Thomas Griffiths, Mark Steyvers, and Padhraic Smyth. The author-topic model for authors and documents. In *UAI '04: Proceedings of the 20th conference on Uncertainty in artificial intelligence*, pages 487–494, Arlington, Virginia, United States, 2004. AUAI Press.

[8] D. Newman and S. Block. Probabilistic topic decomposition of an eighteenth-century american newspaper. *J. Am. Soc. Inf. Sci. Technol.*, 57(6):753–767, 2006.

[9] Y. W. Teh, M. I. Jordan, M. J. Beal, and D. M. Blei. Hierarchical Dirichlet processes. *Journal of the American Statistical Association*, 101(476):1566–1581, 2006.

[10] T. L. Griffiths and M. Steyvers. Finding scientific topics. *Proc Natl Acad Sci U S A*, 101 Suppl 1:5228–5235, April 2004.

[11] A. Schröter, T. Zimmermann, R. Premraj, and A. Zeller. If your bug database could talk.... In *Proceedings of the 5th International Symposium on Empirical Software Engineering, Volume II: Short Papers and Posters*, pages 18–20, September 2006.

[12] E. Brill. Some advances in transformation-based part of speech tagging. In *National Conference on Artificial Intelligence*, pages 722–727, 1994.

[13] E. Ukkonen. Approximate string-matching with q-grams and maximal matches. *Theor. Comput. Sci.*, 92(1):191–211, 1992.

[14] G. Kiczales, J. Lamping, A. Menhdhekar, C. Maeda, C. Lopes, J. Loingtier, and J. Irwin. Aspect-oriented programming. In Mehmet Akşit and Satoshi Matsuoka, editors, *Proceedings European Conference on Object-Oriented Programming*, volume 1241, pages 220–242. Springer-Verlag, Berlin, Heidelberg, and New York, 1997.

[15] R. Motwani L. Page, S. Brin and T. Winograd. The pagerank citation ranking: Bringing order to the web. *Stanford Digital Library working paper SIDL-WP-1999-0120 of 11/11/1999 (see: http://dbpubs.stanford.edu/pub/1999-66)*.
